# Triangle Fixing Algorithms for the Metric Nearness Problem

**Inderjit S. Dhillon**　　**Suvrit Sra**
Dept. of Computer Sciences
The Univ. of Texas at Austin
Austin, TX 78712.
{inderjit,suvrit}@cs.utexas.edu

**Joel A. Tropp**
Dept. of Mathematics
The Univ. of Michigan at Ann Arbor
Ann Arbor, MI, 48109.
jtropp@umich.edu

## Abstract

Various problems in machine learning, databases, and statistics involve pairwise distances among a set of objects. It is often desirable for these distances to satisfy the properties of a metric, especially the triangle inequality. Applications where metric data is useful include clustering, classification, metric-based indexing, and approximation algorithms for various graph problems. This paper presents the *Metric Nearness Problem*: Given a dissimilarity matrix, find the "nearest" matrix of distances that satisfy the triangle inequalities. For $\ell_p$ nearness measures, this paper develops efficient *triangle fixing* algorithms that compute globally optimal solutions by exploiting the inherent structure of the problem. Empirically, the algorithms have time and storage costs that are linear in the number of triangle constraints. The methods can also be easily parallelized for additional speed.

## 1   Introduction

Imagine that a lazy graduate student has been asked to measure the pairwise distances among a group of objects in a metric space. He does not complete the experiment, and he must figure out the remaining numbers before his adviser returns from her conference. Obviously, all the distances need to be consistent, but the student does not know very much about the space in which the objects are embedded. One way to solve his problem is to find the "nearest" complete set of distances that satisfy the triangle inequalities. This procedure respects the measurements that have already been taken while forcing the missing numbers to behave like distances.

More charitably, suppose that the student has finished the experiment, but—measurements being what they are—the numbers do not satisfy the triangle inequality. The student knows that they must represent distances, so he would like to massage the data so that it corresponds with his *a priori* knowledge. Once again, the solution seems to require the "nearest" set of distances that satisfy the triangle inequalities.

Matrix nearness problems [6] offer a natural framework for developing this idea. If there are $n$ points, we may collect the measurements into an $n \times n$ symmetric matrix whose $(j, k)$ entry represents the dissimilarity between the $j$-th and $k$-th points. Then, we seek to approximate this matrix by another whose entries satisfy the triangle inequalities. That is,

$m_{ik} \leq m_{ij} + m_{jk}$ for every triple $(i, j, k)$. Any such matrix will represent the distances among $n$ points in some metric space. We calculate approximation error with a distortion measure that depends on how the corrected matrix should relate to the input matrix. For example, one might prefer to change a few entries significantly or to change all the entries a little.

We call the problem of approximating general dissimilarity data by metric data the *Metric Nearness (MN) Problem*. This simply stated problem has not previously been studied, although the literature does contain some related topics (see Section 1.1). This paper presents a formulation of the Metric Nearness Problem (Section 2), and it shows that every locally optimal solution is globally optimal. To solve the problem we present triangle-fixing algorithms that take advantage of its structure to produce globally optimal solutions. It can be computationally prohibitive, both in time and storage, to solve the MN problem without these efficiencies.

## 1.1 Related Work

The Metric Nearness (MN) problem is novel, but the literature contains some related work.

The most relevant research appears in a recent paper of Roth et al. [11]. They observe that machine learning applications often require metric data, and they propose a technique for metrizing dissimilarity data. Their method, constant-shift embedding, increases all the dissimilarities by an equal amount to produce a set of Euclidean distances (i.e., a set of numbers that can be realized as the pairwise distances among an ensemble of points in a Euclidean space). The size of the translation depends on the data, so the relative and absolute changes to the dissimilarity values can be large. Our approach to metrizing data is completely different. We seek a consistent set of distances that *deviates as little as possible* from the original measurements. In our approach, the resulting set of distances can arise from an arbitrary metric space; we do not restrict our attention to obtaining Euclidean distances. In consequence, we expect metric nearness to provide superior denoising. Moreover, our techniques can also learn distances that are missing entirely.

There is at least one other method for inferring a metric. An article of Xing et al. [12] proposes a technique for learning a Mahalanobis distance for data in $\mathbb{R}^s$. That is, a metric $\text{dist}(\boldsymbol{x}, \boldsymbol{y}) = \sqrt{(\boldsymbol{x} - \boldsymbol{y})^T \boldsymbol{G} (\boldsymbol{x} - \boldsymbol{y})}$, where $\boldsymbol{G}$ is an $s \times s$ positive semi-definite matrix. The user specifies that various pairs of points are similar or dissimilar. Then the matrix $\boldsymbol{G}$ is computed by minimizing the total *squared* distances between similar points while forcing the total distances between dissimilar points to exceed one. The article provides explicit algorithms for the cases where $\boldsymbol{G}$ is diagonal and where $\boldsymbol{G}$ is an arbitrary positive semi-definite matrix. In comparison, the metric nearness problem is not restricted to Mahalanobis distances; it can learn a general discrete metric. It also allows us to use specific distance measurements and to indicate our confidence in those measurements (by means of a weight matrix), rather than forcing a binary choice of "similar" or "dissimilar."

The Metric Nearness Problem may appear similar to metric Multi-Dimensional Scaling (MDS) [8], but we emphasize that the two problems are *distinct*. The MDS problem endeavors to find an ensemble of points in a *prescribed* metric space (usually a Euclidean space) such that the distances between these points are close to the set of input distances. In contrast, the MN problem does not seek to find an embedding. In fact MN does not impose any hypotheses on the underlying space other than requiring it to be a metric space.

The outline of rest of the paper is as follows. Section 2 formally describes the MN problem. In Section 3, we present algorithms that allow us to solve MN problems with $\ell_p$ nearness measures. Some applications and experimental results follow in Section 4. Section 5 discusses our results, some interesting connections, and possibilities for future research.

## 2 The Metric Nearness Problem

We begin with some basic definitions. We define a *dissimilarity matrix* to be a nonnegative, symmetric matrix with zero diagonal. Meanwhile, a *distance matrix* is defined to be a dissimilarity matrix whose entries satisfy the triangle inequalities. That is, $\boldsymbol{M}$ is a distance matrix if and only if it is a dissimilarity matrix and $m_{ik} \leq m_{ij} + m_{jk}$ for every triple of distinct indices $(i, j, k)$. Distance matrices arise from measuring the distances among $n$ points in a pseudo-metric space (i.e., two distinct points can lie at zero distance from each other). A distance matrix contains $N = n(n-1)/2$ free parameters, so we denote the collection of all distance matrices by $\mathscr{M}_N$. The set $\mathscr{M}_N$ is a closed, convex cone.

The metric nearness problem requests a distance matrix $\boldsymbol{M}$ that is closest to a given dissimilarity matrix $\boldsymbol{D}$ with respect to some measure of "closeness." In this work, we restrict our attention to closeness measures that arise from norms. Specifically, we seek a distance matrix $\boldsymbol{M}$ so that,

$$\boldsymbol{M} \in \left\{ \underset{\boldsymbol{X} \in \mathscr{M}_N}{\operatorname{argmin}} \left\| \boldsymbol{W} \odot (\boldsymbol{X} - \boldsymbol{D}) \right\| \right\}, \tag{2.1}$$

where $\| \cdot \|$ is a norm, $\boldsymbol{W}$ is a symmetric non-negative weight matrix, and '$\odot$' denotes the elementwise (Hadamard) product of two matrices. The weight matrix reflects our confidence in the entries of $\boldsymbol{D}$. When each $d_{ij}$ represents a measurement with variance $\sigma_{ij}^2$, we might set $w_{ij} = 1/\sigma_{ij}^2$. If an entry of $\boldsymbol{D}$ is missing, one can set the corresponding weight to zero.

**Theorem 2.1.** *The function $\boldsymbol{X} \mapsto \left\| \boldsymbol{W} \odot (\boldsymbol{X} - \boldsymbol{D}) \right\|$ always attains its minimum on $\mathscr{M}_N$. Moreover, every local minimum is a global minimum. If, in addition, the norm is strictly convex and the weight matrix has no zeros or infinities off its diagonal, then there is a unique global minimum.*

*Proof.* The main task is to show that the objective function has no directions of recession, so it must attain a finite minimum on $\mathscr{M}_N$. Details appear in [4]. □

It is possible to use any norm in the metric nearness problem. We further restrict our attention to the $\ell_p$ norms. The associated Metric Nearness Problems are

$$\min_{\boldsymbol{X} \in \mathscr{M}_N} \left[ \sum_{j \neq k} \left| w_{jk} (x_{jk} - d_{jk}) \right|^p \right]^{1/p} \qquad \text{for } 1 \leq p < \infty, \text{ and} \tag{2.2}$$

$$\min_{\boldsymbol{X} \in \mathscr{M}_N} \max_{j \neq k} \left| w_{jk} (x_{jk} - d_{jk}) \right| \qquad \text{for } p = \infty. \tag{2.3}$$

Note that the $\ell_p$ norms are strictly convex for $1 < p < \infty$, and therefore the solution to (2.2) is unique. There is a basic intuition for choosing $p$. The $\ell_1$ norm gives the absolute sum of the (weighted) changes to the input matrix, while the $\ell_\infty$ only reflects the maximum absolute change. The other $\ell_p$ norms interpolate between these extremes. Therefore, a small value of $p$ typically results in a solution that makes a few large changes to the original data, while a large value of $p$ typically yields a solution with many small changes.

## 3 Algorithms

This section describes efficient algorithms for solving the Metric Nearness Problems (2.2) and (2.3). For ease of exposition, we assume all weights to equal one. At first, it may appear that one should use quadratic programming (QP) software when $p = 2$, linear programming (LP) software when $p = 1$ or $p = \infty$, and convex programming software for the remaining $p$. It turns out that the time and storage requirements of this approach can be prohibitive. An efficient algorithm must exploit the structure of the triangle inequalities. In this paper, we develop one such approach, which may be viewed as a *triangle-fixing*

*algorithm*. This method examines each triple of points in turn and optimally enforces any triangle inequality that fails. (The definition of "optimal" depends on the $\ell_p$ nearness measure.) By introducing appropriate corrections, we can ensure that this iterative algorithm converges to a globally optimal solution of MN.

**Notation.**   We must introduce some additional notation before proceeding. To each matrix $\boldsymbol{X}$ of dissimilarities or distances, we associate the vector $\boldsymbol{x}$ formed by stacking the columns of the lower triangle, left to right. We use $x_{ij}$ to refer to the $(i, j)$ entry of the matrix as well as the corresponding component of the vector. Define a constraint matrix $\boldsymbol{A}$ so that $\boldsymbol{M}$ is a distance matrix if and only if $\boldsymbol{Am} \leq \boldsymbol{0}$. Note that each row of $\boldsymbol{A}$ contains three nonzero entries, $+1$, $-1$, and $-1$.

### 3.1   MN for the $\ell_2$ norm

We first develop a triangle-fixing algorithm for solving (2.2) with respect to the $\ell_2$ norm. This case turns out to be the simplest and most illuminating case. It also plays a pivotal role in the algorithms for the $\ell_1$ and $\ell_\infty$ MN problems.

Given a dissimilarity vector $\boldsymbol{d}$, we wish to find its orthogonal projection $\boldsymbol{m}$ onto the cone $\mathscr{M}_N$. Let us introduce an auxiliary variable $\boldsymbol{e} = \boldsymbol{m} - \boldsymbol{d}$ that represents the changes to the original distances. We also define $\boldsymbol{b} = -\boldsymbol{Ad}$. The negative entries of $\boldsymbol{b}$ indicate how much each triangle inequality is violated. The problem becomes

$$\begin{aligned} \min_{\boldsymbol{e}} \; &\|\boldsymbol{e}\|_2, \\ \text{subject to } &\boldsymbol{Ae} \leq \boldsymbol{b}. \end{aligned} \tag{3.1}$$

After finding the minimizer $\boldsymbol{e}^\star$, we can use the relation $\boldsymbol{m}^\star = \boldsymbol{d} + \boldsymbol{e}^\star$ to recover the optimal distance vector.

Here is our approach. We initialize the vector of changes to zero ($\boldsymbol{e} = \boldsymbol{0}$), and then we begin to cycle through the triangles. Suppose that the $(i, j, k)$ triangle inequality is violated, i.e., $e_{ij} - e_{jk} - e_{ki} > b_{ijk}$. We wish to remedy this violation by making an $\ell_2$-minimal adjustment of $e_{ij}$, $e_{jk}$, and $e_{ki}$. In other words, the vector $\boldsymbol{e}$ is projected orthogonally onto the constraint set $\{\boldsymbol{e}' : e'_{ij} - e'_{jk} - e'_{ki} \leq b_{ijk}\}$. This is tantamount to solving

$$\begin{aligned} \min_{\boldsymbol{e}'} \; &\tfrac{1}{2}\big[(e'_{ij} - e_{ij})^2 + (e'_{jk} - e_{jk})^2 + (e'_{ki} - e_{ki})^2\big], \\ \text{subject to } \; &e'_{ij} - e'_{jk} - e'_{ki} = b_{ijk}. \end{aligned} \tag{3.2}$$

It is easy to check that the solution is given by

$$e'_{ij} \leftarrow e_{ij} - \mu_{ijk}, \qquad e'_{jk} \leftarrow e_{jk} + \mu_{ijk}, \qquad \text{and} \qquad e'_{ki} \leftarrow e_{ki} + \mu_{ijk}, \tag{3.3}$$

where $\mu_{ijk} = \tfrac{1}{3}(e_{ij} - e_{jk} - e_{ki} - b_{ijk}) > 0$. Only three components of the vector $\boldsymbol{e}$ need to be updated. The updates in (3.3) show that the largest edge weight in the triangle is decreased, while the other two edge weights are increased.

In turn, we fix each violated triangle inequality using (3.3). We must also introduce a correction term to guide the algorithm to the global minimum. The corrections have a simple interpretation in terms of the dual of the minimization problem (3.1). Each dual variable corresponds to the violation in a single triangle inequality, and each individual correction results in a decrease in the violation. We continue until no triangle receives a significant update.

Algorithm 3.1 displays the complete iterative scheme that performs triangle fixing along with appropriate corrections.

**Algorithm 3.1:** Triangle Fixing For $\ell_2$ norm.

---
TRIANGLE_FIXING($\boldsymbol{D}$, $\epsilon$)
**Input:** Input dissimilarity matrix $\boldsymbol{D}$, tolerance $\epsilon$
**Output:** $\boldsymbol{M} = \text{argmin}_{\boldsymbol{X} \in \mathcal{M}_N} \|\boldsymbol{X} - \boldsymbol{D}\|_2$.
**for** $1 \le i < j < k \le n$
$\quad (z_{ijk}, z_{jki}, z_{kij}) \leftarrow 0$  {Initialize correction terms}
**for** $1 \le i < j \le n$
$\quad e_{ij} \leftarrow 0$ $\qquad\qquad$ {Initial error values for each dissimilarity $d_{ij}$}
$\delta \leftarrow 1 + \epsilon$ $\qquad\qquad$ {Parameter for testing convergence}
**while** $(\delta > \epsilon)$ $\qquad$ {convergence test}
$\quad$ **foreach** triangle $(i, j, k)$
$\qquad b \leftarrow d_{ki} + d_{jk} - d_{ij}$
$\qquad \mu \leftarrow \frac{1}{3}(e_{ij} - e_{jk} - e_{ki} - b)$ $\qquad\qquad\qquad\qquad\qquad\qquad$ $(\star)$
$\qquad \theta \leftarrow \min\{-\mu, z_{ijk}\}$ $\qquad$ {Stay within half-space of constraint}
$\qquad e_{ij} \leftarrow e_{ij} - \theta, \, e_{jk} \leftarrow e_{jk} + \theta, \, e_{ki} \leftarrow e_{ki} + \theta$ $\qquad\quad$ $(\star\star)$
$\qquad z_{ijk} \leftarrow z_{ijk} - \theta$ $\qquad\qquad$ {Update correction term}
$\quad$ **end** *foreach*
$\quad \delta \leftarrow$ sum of changes in the $\boldsymbol{e}$ values
**end** *while*
**return** $\boldsymbol{M} = \boldsymbol{D} + \boldsymbol{E}$

---

**Remark:** Algorithm 3.1 is an efficient adaptation of Bregman's method [1]. By itself, Bregman's method would suffer the same storage and computation costs as a general convex optimization algorithm. Our triangle fixing operations allow us to compactly represent and compute the intermediate variables required to solve the problem. The correctness and convergence properties of Algorithm 3.1 follow from those of Bregman's method. Furthermore, our algorithms are very easy to implement.

## 3.2 MN for the $\ell_1$ and $\ell_\infty$ norms

The basic triangle fixing algorithm succeeds only when the norm used in (2.2) is strictly convex. Hence, it cannot be applied directly to the $\ell_1$ and $\ell_\infty$ cases. These require a more sophisticated approach.

First, observe that the problem of minimizing the $\ell_1$ norm of the changes can be written as an LP:

$$\min_{\boldsymbol{e},\boldsymbol{f}} \boldsymbol{0}^T \boldsymbol{e} + \boldsymbol{1}^T \boldsymbol{f}$$
$$\text{subject to } \boldsymbol{Ae} \le \boldsymbol{b}, \quad -\boldsymbol{e} - \boldsymbol{f} \le \boldsymbol{0}, \quad \boldsymbol{e} - \boldsymbol{f} \le \boldsymbol{0}. \tag{3.4}$$

The auxiliary variable $\boldsymbol{f}$ can be interpreted as the absolute value of $\boldsymbol{e}$. Similarly, minimizing the $\ell_\infty$ norm of the changes can be accomplished with the LP

$$\min_{\boldsymbol{e},\zeta} \boldsymbol{0}^T \boldsymbol{e} + \zeta$$
$$\text{subject to } \boldsymbol{Ae} \le \boldsymbol{b}, \quad -\boldsymbol{e} - \zeta\boldsymbol{1} \le \boldsymbol{0}, \quad \boldsymbol{e} - \zeta\boldsymbol{1} \le \boldsymbol{0}. \tag{3.5}$$

We interpret $\zeta = \|\boldsymbol{e}\|_\infty$.

Solving these linear programs using standard software can be prohibitively expensive because of the large number of constraints. Moreover, the solutions are not unique because the $\ell_1$ and $\ell_\infty$ norms are not strictly convex. Instead, we replace the LP by a quadratic program (QP) that is strictly convex and returns the solution of the LP that has minimum $\ell_2$-norm. For the $\ell_1$ case, we have the following result.

**Theorem 3.1** ($\ell_1$ **Metric Nearness**). *Let $\boldsymbol{z} = [\boldsymbol{e}; \boldsymbol{f}]$ and $\boldsymbol{c} = [\boldsymbol{0}; \boldsymbol{1}]$ be partitioned conformally. If (3.4) has a solution, then there exists a $\lambda_0 > 0$, such that for all $\lambda \leq \lambda_0$,*

$$\operatorname*{argmin}_{\boldsymbol{z} \in Z} \|\boldsymbol{z} + \lambda^{-1}\boldsymbol{c}\|_2 \quad = \quad \operatorname*{argmin}_{\boldsymbol{z} \in Z^\star} \|\boldsymbol{z}\|_2, \tag{3.6}$$

*where $Z$ is the feasible set for (3.4) and $Z^\star$ is the set of optimal solutions to (3.4). The minimizer of (3.6) is unique.*

Theorem 3.1 follows from a result of Mangasarian [9, Theorem 2.1-a-i]. A similar theorem may be stated for the $\ell_\infty$ case.

The QP (3.6) can be solved using an augmented triangle-fixing algorithm since the majority of the constraints in (3.6) are triangle inequalities. As in the $\ell_2$ case, the triangle constraints are enforced using (3.3). Each remaining constraint is enforced by computing an orthogonal projection onto the corresponding halfspace. We refer the reader to [5] for the details.

### 3.3  MN for $\ell_p$ norms ($1 < p < \infty$)

Next, we explain how to use triangle fixing to solve the MN problem for the remaining $\ell_p$ norms, $1 < p < \infty$. The computational costs are somewhat higher because the algorithm requires solving a nonlinear equation. The problem may be phrased as

$$\min_{\boldsymbol{e}} \; \frac{1}{p} \|\boldsymbol{e}\|_p^p \qquad \text{subject to} \qquad \boldsymbol{A}\boldsymbol{e} \leq \boldsymbol{b}. \tag{3.7}$$

To enforce a triangle constraint optimally in the $\ell_p$ norm, we need to compute a projection of the vector $\boldsymbol{e}$ onto the constraint set. Define $\varphi(\boldsymbol{x}) = \frac{1}{p} \|\boldsymbol{x}\|_p^p$, and note that $(\nabla\varphi(\boldsymbol{x}))_i = \operatorname{sgn}(x_i) |x_i|^{p-1}$. The projection of $\boldsymbol{e}$ onto the $(i, j, k)$ violating constraint is the solution of

$$\min_{\boldsymbol{e}'} \; \varphi(\boldsymbol{e}') - \varphi(\boldsymbol{e}) - \langle \nabla\varphi(\boldsymbol{e}), \boldsymbol{e}' - \boldsymbol{e} \rangle \quad \text{subject to} \qquad \boldsymbol{a}_{ijk}^T \boldsymbol{e}' = b_{ijk},$$

where $\boldsymbol{a}_{ijk}$ is the row of the constraint matrix corresponding to the triangle inequality $(i, j, k)$. The projection may be determined by solving

$$\nabla\varphi(\boldsymbol{e}') = \nabla\varphi(\boldsymbol{e}) + \mu_{ijk}\, \boldsymbol{a}_{ijk} \qquad \text{so that} \qquad \boldsymbol{a}_{ijk}^T \boldsymbol{e}' = b_{ijk}. \tag{3.8}$$

Since $\boldsymbol{a}_{ijk}$ has only three nonzero entries, we see that $\boldsymbol{e}$ only needs to be updated in three components. Therefore, in Algorithm 3.1 we may replace ($\star$) by an appropriate numerical computation of the parameter $\mu_{ijk}$ and replace ($\star\star$) by the computation of the new value of $\boldsymbol{e}$. Further details are available in [5].

## 4  Applications and Experiments

Replacing a general graph (dissimilarity matrix) by a metric graph (distance matrix) can enable us to use efficient approximation algorithms for NP-Hard graph problems (MAX-CUT clustering) that have guaranteed error for metric data, for example, see [7]. The error from MN will carry over to the graph problem, while retaining the bounds on total error incurred. As an example, constant factor approximation algorithms for MAX-CUT exist for metric graphs [3], and can be used for clustering applications. See [4] for more details.

Applications that use dissimilarity values, such as clustering, classification, searching, and indexing, could potentially be sped up if the data is metric. MN is a natural candidate for enforcing metric properties on the data to permit these speedups.

We were originally motivated to formulate and solve MN by a problem that arose in connection with biological databases [13]. This problem involves approximating mPAM matrices,

which are a derivative of mutation probability matrices [2] that arise in protein sequencing. They represent a certain measure of dissimilarity for an application in protein sequencing. Owing to the manner in which these matrices are formed, they tend not to be distance matrices. Query operations in biological databases have the potential to be dramatically sped up if the data were metric (using a metric based indexing scheme). Thus, one approach is to find the nearest distance matrix to each mPAM matrix and use that approximation in the metric based indexing scheme.

We approximated various mPAM matrices by their nearest distance matrices. The relative errors of the approximations $\|D - M\|/\|D\|$ are reported in Table 1.

Table 1: Relative errors for mPAM dataset ($\ell_1, \ell_2, \ell_\infty$ nearness, respectively)

| Dataset | $\frac{\|D-M\|_1}{\|D\|_1}$ | $\frac{\|D-M\|_2}{\|D\|_2}$ | $\frac{\|D-M\|_\infty}{\|D\|_\infty}$ |
|---|---|---|---|
| mPAM50 | 0.339 | 0.402 | 0.278 |
| mPAM100 | 0.142 | 0.231 | 0.206 |
| mPAM150 | 0.054 | 0.121 | 0.151 |
| mPAM250 | 0.004 | 0.025 | 0.042 |
| mPAM300 | 0.002 | 0.017 | 0.056 |

## 4.1 Experiments

The MN problem has an input of size $N = n(n-1)/2$, and the number of constraints is roughly $N^{3/2}$. We ran experiments to ascertain the empirical behavior of the algorithm. Figure 1 shows log–log plots of the running time of our algorithms for solving the $\ell_1$

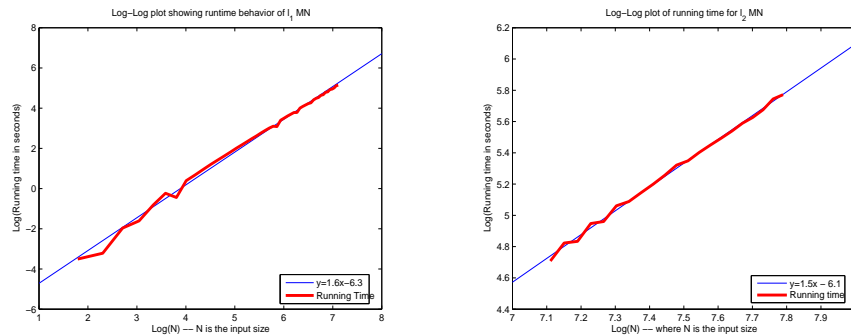

Figure 1: Running time for $\ell_1$ and $\ell_2$ norm solutions (plots have different scales).

and $\ell_2$ Metric Nearness Problems. Note that the time cost appears to be $O(N^{3/2})$, which is *linear* in the number of constraints. The results plotted in the figure were obtained by executing the algorithms on random dissimilarity matrices. The procedure was halted when the distance values changed less than $10^{-3}$ from one iteration to the next. For both problems, the results were obtained with a simple MATLAB implementation. Nevertheless, this basic version outperforms MATLAB's optimization package by one or two orders of magnitude (depending on the problem), while numerically achieving similar results. A more sophisticated (C or parallel) implementation could improve the running time even more, which would allow us to study larger problems.

## 5 Discussion

In this paper, we have introduced the Metric Nearness problem, and we have developed algorithms for solving it for $\ell_p$ nearness measures. The algorithms proceed by fixing violated

triangles in turn, while introducing correction terms to guide the algorithm to the global optimum. Our experiments suggest that the algorithms require $O(N^{3/2})$ time, where $N$ is the total number of distances, so it is linear in the number of constraints. An open problem is to obtain an algorithm with better computational complexity.

Metric Nearness is a rich problem. It can be shown that a special case (allowing only decreases in the dissimilarities) is *identical* with the All Pairs Shortest Path problem [10]. Thus one may check whether the $N$ distances satisfy metric properties in $O(\text{APSP})$ time. However, we are not aware if this is a lower bound.

It is also possible to incorporate other types of linear and convex constraints into the Metric Nearness Problem. Some other possibilities include putting box constraints on the distances ($l \leq m \leq u$), allowing $\lambda$ triangle inequalities ($m_{ij} \leq \lambda_1 m_{ik} + \lambda_2 m_{kj}$), or enforcing order constraints ($d_{ij} < d_{kl}$ implies $m_{ij} < m_{kl}$).

We plan to further investigate the application of MN to other problems in data mining, machine learning, and database query retrieval.

### Acknowledgments

This research was supported by NSF grant CCF-0431257, NSF Career Award ACI-0093404, and NSF-ITR award IIS-0325116.

## References

[1] Y. Censor and S. A. Zenios. *Parallel Optimization: Theory, Algorithms, and Applications*. Numerical Mathematics and Scientific Computation. OUP, 1997.

[2] M. O. Dayhoff, R. M. Schwarz, and B. C. Orcutt. A model of evolutionary change in proteins. *Atlas of Protein Sequence and Structure*, 5(Suppl. 3), 1978.

[3] W. F. de la Vega and C. Kenyon. A randomized approximation scheme for Metric MAX-CUT. *J. Comput. Sys. and Sci.*, 63:531–541, 2001.

[4] I. S. Dhillon, S. Sra, and J. A. Tropp. The Metric Nearness Problems with Applications. Tech. Rep. TR-03-23, Comp. Sci. Univ. of Texas at Austin, 2003.

[5] I. S. Dhillon, S. Sra, and J. A. Tropp. Triangle Fixing Algorithms for the Metric Nearness Problem. Tech. Rep. TR-04-22, Comp. Sci., Univ. of Texas at Austin, 2004.

[6] N. J. Higham. Matrix nearness problems and applications. In M. J. C. Gower and S. Barnett, editors, *Applications of Matrix Theory*, pages 1–27. Oxford University Press, 1989.

[7] P. Indyk. Sublinear time algorithms for metric space problems. In *31st Symposium on Theory of Computing*, pages 428–434, 1999.

[8] J. B. Kruskal and M. Wish. *Multidimensional Scaling*. Number 07-011. Sage Publications, 1978. Series: Quantitative Applications in the Social Sciences.

[9] O. L. Mangasarian. Normal solutions of linear programs. *Mathematical Programming Study*, 22:206–216, 1984.

[10] C. G. Plaxton. Personal Communication, 2003–2004.

[11] V. Roth, J. Laub, J. M. Buhmann, and K.-R. Müller. Going metric: Denoising pariwise data. In S. Becker, S. Thrun, and K. Obermayer, editors, *Advances in Neural Information Processing Systems (NIPS) 15*, 2003.

[12] E. P. Xing, A. Y. Ng, M. I. Jordan, and S. Russell. Distance metric learning, with application to clustering with side constraints. In S. Becker, S. Thrun, and K. Obermayer, editors, *Advances in Neural Information Processing Systems (NIPS) 15*, 2003.

[13] W. Xu and D. P. Miranker. A metric model of amino acid substitution. *Bioinformatics*, 20(0):1–8, 2004.
